# An Information Theoretic Framework for Eukaryotic Gradient Sensing

**Joseph M. Kimmel**$^*$  and  **Richard M. Salter**$^\dagger$
joekimmel@uchicago.edu, rms@cs.oberlin.edu
Computer Science Program
Oberlin College
Oberlin, Ohio 44074

**Peter J. Thomas**$^\ddagger$
peter.j.thomas@case.edu
Departments of Mathematics, Biology and Cognitive Science
Case Western Reserve University
Cleveland, Ohio 44106

## Abstract

Chemical reaction networks by which individual cells gather and process information about their chemical environments have been dubbed "signal transduction" networks. Despite this suggestive terminology, there have been few attempts to analyze chemical signaling systems with the quantitative tools of information theory. Gradient sensing in the social amoeba *Dictyostelium discoideum* is a well characterized signal transduction system in which a cell estimates the direction of a source of diffusing chemoattractant molecules based on the spatiotemporal sequence of ligand-receptor binding events at the cell membrane. Using Monte Carlo techniques (MCell) we construct a simulation in which a collection of individual ligand particles undergoing Brownian diffusion in a three-dimensional volume interact with receptors on the surface of a static amoeboid cell. Adapting a method for estimation of spike train entropies described by Victor (originally due to Kozachenko and Leonenko), we estimate lower bounds on the mutual information between the transmitted signal (direction of ligand source) and the received signal (spatiotemporal pattern of receptor binding/unbinding events). Hence we provide a quantitative framework for addressing the question: how much could the cell know, and when could it know it? We show that the time course of the mutual information between the cell's surface receptors and the (unknown) gradient direction is consistent with experimentally measured cellular response times. We find that the acquisition of directional information depends strongly on the time constant at which the intracellular response is filtered.

## 1    Introduction: gradient sensing in eukaryotes

Biochemical signal transduction networks provide the computational machinery by which neurons, amoebae or other single cells sense and react to their chemical environments. The precision of this chemical sensing is limited by fluctuations inherent in reaction and diffusion processes involving a

---

$^*$Current address: Computational Neuroscience Graduate Program, The University of Chicago.

$^\dagger$Oberlin Center for Computation and Modeling, http://occam.oberlin.edu/.

$^\ddagger$To whom correspondence should be addressed.   http://www.case.edu/artsci/math/thomas/thomas.html; Oberlin College Research Associate.

finite quantity of molecules [1, 2]. The theory of communication provides a framework that makes explicit the noise dependence of chemical signaling. For example, in any reaction $A + B \rightarrow C$, we may view the time varying reactant concentrations $A(t)$ and $B(t)$ as input signals to a noisy channel, and the product concentration $C(t)$ as an output signal carrying information about $A(t)$ and $B(t)$. In the present study we show that the mutual information between the (known) state of the cell's surface receptors and the (unknown) gradient direction follows a time course consistent with experimentally measured cellular response times, reinforcing earlier claims that information theory can play a role in understanding biochemical cellular communication [3, 4].

*Dictyostelium* is a soil dwelling amoeba that aggregates into a multicellular form in order to survive conditions of drought or starvation. During aggregation individual amoebae perform *chemotaxis*, or chemically guided movement, towards sources of the signaling molecule cAMP, secreted by nearby amoebae. Quantitive studies have shown that *Dictyostelium* amoebae can sense shallow, static gradients of cAMP over long time scales ($\sim$30 minutes), and that gradient steepness plays a crucial role in guiding cells [5]. The *chemotactic efficiency* (CE), the population average of the cosine between the cell displacement directions and the true gradient direction, peaks at a cAMP concentration of 25 nanoMolar, similar to the equilibrium constant for the cAMP receptor (the $K_{eq}$ is the concentration of cAMP at which the receptor has a 50% chance of being bound or unbound, respectively). For smaller or larger concentrations the CE dropped rapidly. Nevertheless over long times cells were able (on average) to detect gradients as small as 2% change in [cAMP] per cell length. At an early stage of development when the pattern of chemotactic centers and spirals is still forming, individual amoebae presumably experience an inchoate barrage of weak, noisy and conflicting directional signals. When cAMP binds receptors on a cell's surface, second messengers trigger a chain of subsequent intracellular events including a rapid spatial reorganization of proteins involved in cell motility. Advances in fluorescence microscopy have revealed that the oriented subcellular response to cAMP stimulation is already well underway within two seconds [6, 7]. In order to understand the fundamental limits to communication in this cell signaling process we abstract the problem faced by a cell to that of rapidly identifying the direction of origin of a stimulus gradient superimposed on an existing mean background concentration. We model gradient sensing as an information channel in which an input signal – the direction of a chemical source – is noisily transmitted *via* a gradient of diffusing signaling molecules; and the "received signal" is the spatiotemporal pattern of binding events between cAMP and the cAMP receptors [8]. We neglect downstream intracellular events, which cannot increase the mutual information between the state of the cell and the direction of the imposed extracellular gradient [9].

The analysis of any signal transmission system depends on precise representation of the noise corrupting transmitted signals. We develop a Monte Carlo simulation (MCell, [10, 11]) in which a simulated cell is exposed to a cAMP distribution that evolves from a uniform background to a gradient at low (1 nMol) average concentration. The noise inherent in the communication of a diffusion-mediated signal is accurately represented by this method. Our approach bridges both the transient and the steady state regimes and allows us to estimate the amount of stimulus-related information that is in principle available to the cell through its receptors as a function of time after stimulus initiation. Other efforts to address aspects of cell signaling using the conceptual tools of information theory have considered neurotransmitter release [3] and sensing temporal signals [4], but not gradient sensing in eukaryotic cells.

A typical natural habitat for social amoebae such as *Dictyostelium* is the complex anisotropic three-dimensional matrix of the forest floor. Under experimental conditions cells typically aggregate on a flat two-dimensional surface. We approach the problem of gradient sensing on a sphere, which is both harder and more natural for the ameoba, while still simple enough for us analytically and numerically. Directional data is naturally described using unit vectors in spherical coordinates, but the ameobae receive signals as binding events involving intramembrane protein complexes, so we have developed a method for projecting the ensemble of receptor bindings onto coordinates in $\mathbb{R}^3$. In loose analogy with the chemotactic efficiency [5], we compare the projected directional estimate with the true gradient direction represented as a unit vector on $\mathbb{S}^2$. Consistent with observed timing of the cell's response to cAMP stimulation, we find that the directional signal converges quickly enough for the cell to make a decision about which direction to move within the first two seconds following stimulus onset.

## 2  Methods

### 2.1  Monte Carlo simulations

Using MCell and DReAMM [10, 11] we construct a spherical cell (radius $R = 7.5\mu$m, [12]) centered in a cubic volume (side length $L = 30\mu$m). $N = 980$ triangular tiles partition the surface (mesh generated by DOME[1]); each contained one cell surface receptor for cAMP with binding rate $k_+ = 4.4 \times 10^7$ sec$^{-1}$M$^{-1}$, first-order cAMP unbinding rate $k_- = 1.1$ sec$^{-1}$ [12] and $K_{eq} = k_-/k_+ = 25$nMol cAMP.

We established a baseline concentration of approximately 1nMol by releasing a cAMP bolus at time 0 inside the cube with zero-flux boundary conditions imposed on each wall. At t = 2 seconds we introduced a steady flux at the $x = -L/2$ wall of 1 molecule of cAMP per square micron per msec, adding signaling molecules from the left. Simultaneously, the $x = +L/2$ wall of the cube assumes absorbing boundary conditions. The new boundary conditions lead (at equilibrium) to a linear gradient of 2 nMol/$30\mu m$, ranging from $\approx 2.0$ nMol at the flux source wall to $\approx 0$ nMol at the absorbing wall (see Figure 1); the concentration profile approaches this new steady state with time constant of approximately 1.25 msec. Sampling boxes centered along the planes $x = \pm 13.5\mu$m measured the local concentration, allowing us to validate the expected model behavior.

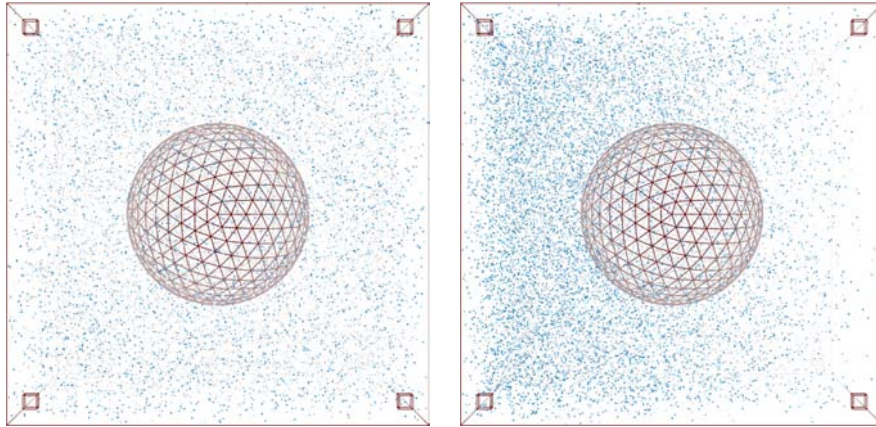

Figure 1: Gradient sensing simulations performed with *MCell* (a Monte Carlo simulator of cellular microphysiology, http://www.mcell.cnl.salk.edu/) and rendered with *DReAMM* (Design, Render, and Animate *MCell* Models, http://www.mcell.psc.edu/). The model cell comprised a sphere triangulated with 980 tiles with one cAMP receptor per tile. Cell radius $R = 7.5\mu$m; cube side $L = 30\mu$m. **Left:** Initial equilibrium condition, before imposition of gradient. [cAMP] $\approx$ 1nMol (c. 15,000 molecules in the volume outside the sphere). **Right:** Gradient condition after transient (c. 15,000 molecules; see Methods for details).

### 2.2  Analysis

#### 2.2.1  Assumptions

We make the following assumptions to simplify the analysis of the distribution of receptor activities at equilibrium, whether pre- or post-stimulus onset:

1. *Independence.* At equilibrium, the state of each receptor (bound *vs* unbound) is independent of the states of the other receptors.

2. *Linear Gradient.* At equilibrium under the imposed gradient condition, the concentration of ligand molecule varies linearly with position along the gradient axis.

3. *Symmetry.*

(a) *Rotational equivariance of receptor activities.* In the absence of an applied gradient signal, the probability distribution describing the receptor states is equivariant with respect to arbitrary rotations of the sphere.

(b) *Rotational invariance of gradient direction.* The imposed gradient seen by a model cell is equally likely to be coming from any direction; therefore the gradient direction vector is uniformly distributed over $\mathbb{S}^2$.

(c) *Axial equivariance about the gradient direction.* Once a gradient direction is imposed, the probability distribution describing receptor states is rotationally equivariant with respect to rotations about the axis parallel with the gradient.

Berg and Purcell [1] calculate the inaccuracy in concentration estimates due to nonindependence of adjacent receptors; for our parameters (effective receptor radius = 5nm, receptor spacing $\sim 1\mu$m) the fractional error in estimating concentration differences due to receptor nonindependence is negligible ($\lesssim 10^{-11}$) [1, 2].

Because we fix receptors to be in 1:1 correspondence with surface tiles, spherical symmetry and uniform distribution of the receptors are only approximate. The gradient signal communicated *via* diffusion does not involve sharp spatial changes on the scale of the distance between nearby receptors, therefore spherical symmetry and uniform identical receptor distribution are good analytic approximations of the model configuration. By *rotational equivariance* we mean that combining any rotation of the sphere with a corresponding rotation of the indices labeling the $N$ receptors, $\{j = 1, \cdots, N\}$, leads to a statistically indistinguishable distribution of receptor activities. This same spherical symmetry is reflected in the *a priori* distribution of gradient directions, which is uniform over the sphere (with density $1/4\pi$). Spherical symmetry is broken by the gradient signal, which fixes a preferred direction in space. About this axis however, we assume the system retains the rotational symmetry of the cylinder.

### 2.2.2 Mutual information of the receptors

In order to quantify the directional information available to the cell from its surface receptors we construct an explicit model for the receptor states and the cell's estimated direction. We model the receptor states via a collection of random variables $\{B_j\}$ and develop an expression for the entropy of $\{B_j\}$. Then in section 2.2.3 we present a method for projecting a temporally filtered estimated direction, $\hat{g}$, into three (rather than $N$) dimensions.

Let the random variables $\{B_j\}_{j=1}^N$ represent the states of the $N$ cAMP receptors on the cell surface; $B_j = 1$ if the receptor is bound to a molecule of cAMP, otherwise $B_j = 0$. Let $\vec{x}_j \in \mathbb{S}^2$ represent the direction from the center of the center of the cell to the $j^{\text{th}}$ receptor. Invoking assumption 2 above, we take the equilibrium concentration of cAMP at $\vec{x}$ to be $c(\vec{x}|\vec{g}) = a + b(\vec{x} \cdot \vec{g})$ where $\vec{g} \in \mathbb{S}^2$ is a unit vector in the direction of the gradient. The parameter $a$ is the mean concentration over the cell surface, and $b = R|\vec{\nabla}c|$ is half the drop in concentration from one extreme on the cell surface to the other. Before the stimulus begins, the gradient direction is undefined.

It can be shown (see Supplemental Materials) that the entropy of receptor states given a fixed gradient direction $\vec{g}$, $H[\{B_j\}|\vec{g}]$, is given by an integral over the sphere:

$$H[\{B_j\}|\vec{g}] \sim N \int_{\theta=0}^{\pi} \int_{\phi=0}^{2\pi} \Phi\left[\frac{a + b\cos(\theta)}{a + b\cos(\theta) + K_{\text{eq}}}\right] \frac{\sin(\theta)}{4\pi} \, d\phi \, d\theta \quad \text{(as } N \to \infty). \quad (1)$$

On the other hand, if the gradient direction remains *unspecified*, the entropy of receptor states is given by

$$H[\{B_j\}] \sim N\Phi\left[\int_{\theta=0}^{\pi} \int_{\phi=0}^{2\pi} \left(\frac{a + b\cos(\theta)}{a + b\cos(\theta) + K_{\text{eq}}}\right) \frac{\sin(\theta)}{4\pi} \, d\phi \, d\theta\right] \quad \text{(as } N \to \infty), \quad (2)$$

where $\Phi[p] = \left\{ \begin{array}{ll} -(p\log_2(p) + (1-p)\log_2(1-p)), & 0 < p < 1 \\ 0, & p = 0 \text{ or } 1 \end{array} \right\}$ denotes the entropy for a binary random variable with state probabilities $p$ and $(1-p)$.

In both equations (1) and (2), the argument of $\Phi$ is a probability taking values $0 \leq p \leq 1$. In (1) the values of $\Phi$ are averaged over the sphere; in (2) $\Phi$ is evaluated after averaging probabilities. Because

$\Phi[p]$ is convex for $0 \leq p \leq 1$, the integral in equation 1 cannot exceed that in equation 2. Therefore the mutual information upon receiving the signal is nonnegative (as expected):

$$MI[\{B_j\}; \vec{g}] \triangleq H[\{B_j\}] - H[\{B_j\}|\vec{g}] \geq 0.$$

The analytic solution for equation (1) involves the polylogarithm function. For the parameters shown in the simulation ($a = 1.078$ nMol, $b = .512$ nMol, $K_{\text{eq}} = 25$ nMol), the mutual information with 980 receptors is 2.16 bits. As one would expect, the mutual information peaks when the mean concentration is close to the $K_{\text{eq}}$ of the receptor, exceeding 16 bits when $a = 25, b = 12.5$ and $K_{\text{eq}} = 25$ (nMol).

### 2.2.3 Dimension reduction

The estimate obtained above does not give tell us how quickly the directional information available to the cell evolves over time. Direct estimate of the mutual information from stochastic simulations is impractical because the aggregate random variables occupy a 980 dimensional space that a limited number of simulation runs cannot sample adequately. Instead, we construct a deterministic function from the set of 980 time courses of the receptors, $\{B_j(t)\}$, to an aggregate directional estimate in $\mathbb{R}^3$. Because of the cylindrical symmetry inherent in the system, our directional estimator $\hat{g}$ is an unbiased estimator of the true gradient direction $\vec{g}$. The estimator $\hat{g}(t)$ may be thought of as representing a downstream chemical process that accumulates directional information and decays with some time constant $\tau$. Let $\{\vec{x}_j\}_{j=1}^N$ be the spatial locations of the $N$ receptors on the cell's surface. Each vector is associated with a weight $w_j$. Whenever the $j^{\text{th}}$ receptor binds a cAMP molecule, $w_j$ is incremented by one; otherwise $w_j$ decays with time constant $\tau$. We construct an instantaneous estimate of the gradient direction from the linear combination of receptor positions, $\hat{g}_\tau(t) = \sum_{j=1}^N w_j(t)\vec{x}_j$. This procedure reflects the accumulation and reabsorption of intracellular second messengers released from the cell membrane upon receptor binding.

Before the stimulus is applied, the weighted directional estimates $\hat{g}_\tau$ are small in absolute magnitude, with direction uniformly distributed on $\mathbb{S}^2$. In order to determine the information gained as the estimate vector evolves after stimulus application, we wish to determine the change in entropy in an ensemble of such estimates. As the cell gains information about the direction of the gradient signal from its receptors, the entropy of the estimate should decrease, leading to a rise in mutual information. By repeating multiple runs ($M = 600$) of the simulation we obtain samples from the ensemble of direction estimates, given a particular stimulus direction, $\vec{g}$. In the method of Kozachenko and Leonenko [13], adapted for the analysis of neural spike train data by Victor [14] ("KLV method"), the cumulative distribution function is approximated directly from the observed samples, and the entropy is estimated *via* a change of variables transformation (see below). This method may be formulated in vector spaces $\mathbb{R}^d$ for $d > 1$ ([13]), but it is not guaranteed to be unbiased in the multivariate case [15] and has not been extended to curved manifolds such as the sphere. In the present case, however, we may exploit the symmetries inherent in the model (Assumptions 3a-3c) to reduce the empirical entropy estimation problem to one dimension.

Adapting the argument in [14] to the case of spherical data from a distribution with rotational symmetry about a given axis, we obtain an estimate of the entropy based on a series of observations of the angles $\{\theta_1, \cdots, \theta_M\}$ between the estimates $\hat{g}_\tau$ and the true gradient direction $\vec{g}$ (for details, see Supplemental Materials):

$$H \sim \frac{1}{M} \sum_{k=1}^M \left( \log_2(\lambda_k) + \log_2(2(M-1)) + \frac{\gamma}{\log_e(2)} + \log_2(2\pi) + \log_2(\sin(\theta_k)) \right) \quad (3)$$

(as $M \to \infty$) where after sorting the $\theta_k$ in monotonic order, $\lambda_k \triangleq \min(|\theta_k - \theta_{k\pm1}|)$ is the distance between each angle and its nearest neighbor in the sample, and $\gamma$ is the Euler-Mascheroni constant. As shown in Figure 2, this approximation agrees with the analytic result for the uniform distribution, $H_{\text{unif}} = \log_2(4\pi) \approx 3.651$.

## 3  Results

Figure 3 shows the results of $M = 600$ simulation runs. Panel **A** shows the concentration averaged across a set of $1\mu\text{m}^3$ sample boxes, four in the $x = -13.5\mu\text{m}$ plane and four in the $x = +13.5\mu\text{m}$

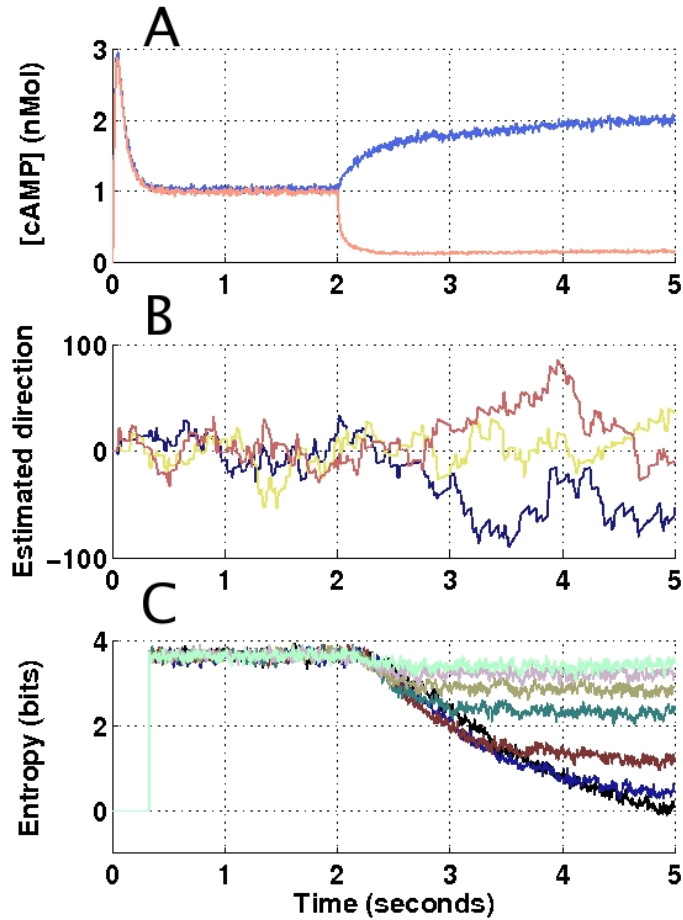

Figure 2: Monte Carlo simulation results and information analysis. **A:** Average concentration profiles along two planes perpendicular to the gradient, at $x = \pm 13.5\mu$m. **B:** Estimated direction vector (x, y, and z components; x = dark blue trace) $\hat{g}_\tau$, $\tau = 500$ msec. **C:** Entropy of the ensemble of directional vector estimates for different values of the intracellular filtering time constant $\tau$. Given the directions of the estimates $\theta_k, \phi_k$ on each of $M$ runs, we calculate the entropy of the ensemble using equation (3). All time constants yield uniformly distributed directional estimates in the pre-stimulus period, $0 \leq t \leq 2$ (sec). After stimulus onset, directional estimates obtained with shorter time constants respond more quickly but achieve smaller gains in mutual information (smaller reductions in entropy). Filtering time constants $\tau$ range from lightest to darkest colors: 20, 50, 100, 200, 500, 1000, 2000 msec.

plane. The initial bolus of cAMP released into the volume at $t = 0$ sec is not uniformly distributed, but spreads out evenly within 0.25 sec. At $t = 2.0$ sec the boundary conditions are changed, causing a gradient to emerge along a realistic time course. Consistent with the analytic solution for the mean concentration (not shown), the concentration approaches equilibrium more rapidly near the absorbing wall (descending trace) than at the imposed flux wall (ascending trace).

Panel **B** shows the evolution of a directional estimate vector $\hat{g}_\tau$ for a single run, with $\tau = 500$ msec. During uniform conditions all vectors fluctuate near the origin. After gradient onset the variance increases and the x component (dark trace) becomes biased towards the gradient source ($\vec{g} = [-1, 0, 0]$) while the y and z components still have a mean of zero. Across all 600 runs the mean of the y and z components remains close to zero, while the mean of the x component systematically departs from zero shortly after stimulus onset (not shown). Hence the directional

estimator is unbiased (as required by symmetry). See Supplemental Materials for the population average of $\hat{g}$.

Panel **C** shows the time course of the entropy of the ensemble of normalized directional estimate vectors $\hat{g}_\tau/|\hat{g}_\tau|$ over $M = 600$ simulations, for intracellular filtering time constants ranging from 20 msec to 2000 msec (light to dark shading), calculated using equation (3). Following stimulus onset, entropy decreases steadily, showing an increase in information available to the amoeba about the direction of the stimulus; the mutual information at a given point in time is the difference between the entropy at that time and before stimulus onset.

For a cell with roughly 1000 receptors the mutual information has increased at most by $\sim 2$ bits of information by one second (for $\tau = 500$ msec), and at most by $\sim 3$ bits of information by two seconds (for $\tau$=1000 or 2000 msec), under our stimulation protocol. A one bit reduction in uncertainty is equivalent to identifying the correct value of the $x$ component (positive versus negative) when the stimulus direction is aligned along the $x$-axis. Alternatively, note that a one bit reduction results in going from the uniform distribution on the sphere to the uniform distribution on one hemisphere. For $\tau \leq 100$ msec, the weighted average with decay time $\tau$ never gains more than one bit of information about the stimulus direction, even at long times. This observation suggests that signaling must involve some chemical components with lifetimes longer than 100 msec. The $\tau = 200$ msec filter saturates after about one second, at $\sim 1$ bit of information gain.

Longer lived second messengers would respond more slowly to changes from the background stimulus distribution, but would provide better more informative estimates over time. The $\tau = 500$ msec estimate gains roughly two bits of information within 1.5 seconds, but not much more over time. Heuristically, we may think of a two bit gain in information as corresponding to the change from a uniform distribution to one covering uniformly covering one quarter of $\mathbb{S}^2$, *i.e.* all points within $\pi/3$ of the true direction. Within two seconds the $\tau = 1000$ msec and $\tau = 2000$ msec weighted averages have each gained approximately three bits of information, equivalent to a uniform distribution covering all points with $0.23\pi$ or $41^o$ of the true direction.

# 4 Discussion & conclusions

Clearly there is an opportunity for more precise control of experimental conditions to deepen our understanding of spatio-temporal information processing at the membranes of gradient-sensitive cells. Efforts in this direction are now using microfluidic technology to create carefully regulated spatial profiles for probing cellular responses [16]. Our results suggest that molecular processes relevant to these responses must have lasting effects $\geq 100$ msec.

We use a static, immobile cell. Could cell motion relative to the medium increase sensitivity to changes in the gradient? No: the *Dictyostelium* velocity required to affect concentration perception is on order $1\text{cm}\,\text{sec}^{-1}$[1], whereas reported velocities are on the order $\mu\text{m}\,\text{sec}^{-1}$[5].

The chemotactic response mechanism is known to begin modifying the cell membrane on the edge facing up the gradient within two seconds after stimulus initiation [7, 6], suggesting that the cell strikes a balance between gathering data and deciding quickly. Indeed, our results show that the reported activation of the G-protein signaling system on the leading edge of a chemotactically responsive cell [7] rises at roughly the same rate as the available chemotactic information. Results such as these ([7, 6]) are obtained by introducing a pipette into the medium near the amoeba; the magnitude and time course of cAMP release are not precisely known, and when estimated the cAMP concentration at the cell surface is over 25 nMol by a full order of magnitude.

Thomson and Kristan [17] show that for discrete probability distributions and for continuous distributions over linear spaces, stimulus discriminability may be better quantified using ideal observer analysis (mean squared error, for continuous variables) than information theory. The machinery of mean squared error (variance, expectation) do not carry over to the case of directional data without fundamental modifications [18]; in particular the notion of mean squared error is best represented by the *mean resultant length* $0 \leq \rho \leq 1$, the expected length of the vector average of a collection of unit vectors representing samples from directional data. A resultant with length $\rho \approx 1$ corresponds to a highly focused probability density function on the sphere. In addition to measuring the mutual information between the gradient direction and an intracellular estimate of direction, we also calculated the time evolution of $\rho$ (see Supplemental Materials.) We find that $\rho$ rapidly approaches 1

and can exceed 0.9, depending on $\tau$. We found that in this case at least the behavior of the mean resultant length and the mutual information are very similar; there is no evidence of discrepancies of the sort described in [17].

We have shown that the mutual information between an arbitrarily oriented stimulus and the directional signal available at the cell's receptors evolves with a time course consistent with observed reaction times of *Dictyostelium* amoeba. Our results reinforce earlier claims that information theory can play a role in understanding biochemical cellular communication.

### Acknowledgments

MCell simulations were run on the Oberlin College Beowulf Cluster, supported by NSF grant CHE-0420717.

## Footnotes

[1]http://nwg.phy.bnl.gov/~bviren/uno/other/

## References

[1] Howard C. Berg and Edward M. Purcell. Physics of chemoreception. *Biophysical Journal*, 20:193, 1977.

[2] William Bialek and Sima Setayeshgar. Physical limits to biochemical signaling. *PNAS*, 102(29):10040–10045, July 19 2005.

[3] S. Qazi, A. Beltukov, and B.A. Trimmer. Simulation modeling of ligand receptor interactions at non-equilibrium conditions: processing of noisy inputs by ionotropic receptors. *Math Biosci.*, 187(1):93–110, Jan 2004.

[4] D. J. Spencer, S. K. Hampton, P. Park, J. P. Zurkus, and P. J. Thomas. The diffusion-limited biochemical signal-relay channel. In S. Thrun, L. Saul, and B. Schölkopf, editors, *Advances in Neural Information Processing Systems 16*. MIT Press, Cambridge, MA, 2004.

[5] P.R. Fisher, R. Merkl, and G. Gerisch. Quantitative analysis of cell motility and chemotaxis in *Dictyostelium discoideum* by using an image processing system and a novel chemotaxis chamber providing stationary chemical gradients. *J. Cell Biology*, 108:973–984, March 1989.

[6] Carole A. Parent, Brenda J. Blacklock, Wendy M. Froehlich, Douglas B. Murphy, and Peter N. Devreotes. G protein signaling events are activated at the leading edge of chemotactic cells. *Cell*, 95:81–91, 2 October 1998.

[7] Xuehua Xu, Martin Meier-Schellersheim, Xuanmao Jiao, Lauren E. Nelson, and Tian Jin. Quantitative imaging of single live cells reveals spatiotemporal dynamics of multistep signaling events of chemoattractant gradient sensing in *dictyostelium*. *Molecular Biology of the Cell*, 16:676–688, February 2005.

[8] Jan Wouter-Rappel, Peter. J Thomas, Herbert Levine, and William F. Loomis. Establishing direction during chemotaxis in eukaryotic cells. *Biophys. J.*, 83:1361–1367, 2002.

[9] T.M. Cover and J.A. Thomas. *Elements of Information Theory*. John Wiley, New York, 1990.

[10] J. R. Stiles, D. Van Helden, T. M. Bartol, E.E. Salpeter, and M. M. Salpeter. Miniature endplate current rise times less than 100 microseconds from improved dual recordings can be modeled with passive acetylcholine diffusion from a synaptic vesicle. *Proc. Natl. Acad. Sci. U.S.A.*, 93(12):5747–52, Jun 11 1996.

[11] J. R. Stiles and T. M. Bartol. *Computational Neuroscience: Realistic Modeling for Experimentalists*, chapter Monte Carlo methods for realistic simulation of synaptic microphysiology using MCell, pages 87–127. CRC Press, Boca Raton, FL, 2001.

[12] M. Ueda, Y. Sako, T. Tanaka, P. Devreotes, and T. Yanagida. Single-molecule analysis of chemotactic signaling in *Dictyostelium* cells. *Science*, 294:864–867, October 2001.

[13] L.F. Kozachenko and N.N. Leonenko. *Probl. Peredachi Inf. [Probl. Inf. Transm.]*, 23(9):95, 1987.

[14] Jonathan D. Victor. Binless strategies for estimation of information from neural data. *Physical Review E*, 66:051903, Nov 11 2002.

[15] Marc M. Van Hulle. Edgeworth approximation of multivariate differential entropy. *Neural Computation*, 17:1903–1910, 2005.

[16] Loling Song, Sharvari M. Nadkarnia, Hendrik U. Bödekera, Carsten Beta, Albert Bae, Carl Franck, Wouter-Jan Rappel, William F. Loomis, and Eberhard Bodenschatz. *Dictyostelium discoideum* chemotaxis: Threshold for directed motion. *Euro. J. Cell Bio*, 85(9-10):981–9, 2006.

[17] Eric E. Thomson and William B. Kristan. Quantifying stimulus discriminability: A comparison of information theory and ideal observer analysis. *Neural Computation*, 17:741–778, 2005.

[18] Kanti V. Mardia and Peter E. Jupp. *Directional Statistics*. John Wiley & Sons, West Sussex, England, 2000.
